# Speech Modelling Using Subspace and EM Techniques

**Gavin Smith**
Cambridge University
Engineering Department
Cambridge CB2 1PZ
England
gas1003@eng.cam.ac.uk

**João FG de Freitas**
Computer Science Division
487 Soda Hall
UC Berkeley
CA 94720-1776, USA.
jfgf@cs.berkeley.edu [1]

**Tony Robinson**
Cambridge University
Engineering Department
Cambridge CB2 1PZ
England
ajr@eng.cam.ac.uk

**Mahesan Niranjan**
Computer Science
Sheffield University
Sheffield. S1 4DP
England
m.niranjan@dcs.shef.ac.uk

## Abstract

The speech waveform can be modelled as a piecewise-stationary linear stochastic state space system, and its parameters can be estimated using an expectation-maximisation (EM) algorithm. One problem is the initialisation of the EM algorithm. Standard initialisation schemes can lead to poor formant trajectories. But these trajectories however are important for vowel intelligibility. The aim of this paper is to investigate the suitability of subspace identification methods to initialise EM.

The paper compares the subspace state space system identification (4SID) method with the EM algorithm. The 4SID and EM methods are similar in that they both estimate a state sequence (but using Kalman filters and Kalman smoothers respectively), and then estimate parameters (but using least-squares and maximum likelihood respectively). The similarity of 4SID and EM motivates the use of 4SID to initialise EM. Also, 4SID is non-iterative and requires no initialisation, whereas EM is iterative and requires initialisation. However 4SID is sub-optimal compared to EM in a probabilistic sense. During experiments on real speech, 4SID methods compare favourably with conventional initialisation techniques. They produce smoother formant trajectories, have greater frequency resolution, and produce higher likelihoods.

# 1 Introduction

This paper models speech using a stochastic state space model, where model parameters are estimated using the expectation-maximisation (EM) technique. One problem is the initialisation of the EM algorithm. Standard initialisation schemes can lead to poor formant trajectories. These trajectories are however important for vowel intelligibility. This paper investigates the suitability of subspace state space system identification (4SID) techniques [10,11], which are popular in system identification, for EM initialisation.

Speech is split into fixed-length, overlapping frames. Overlap encourages temporally smoother parameter transitions between frames. Due to the slow non-stationary behaviour of speech, each frame of speech is assumed quasi-stationary and represented as a linear time-invariant stochastic state space (SS) model.

$$\mathbf{x}_{t+1} = \mathbf{A}\mathbf{x}_t + \mathbf{w}_t \tag{1}$$
$$y_t = \mathbf{c}\mathbf{x}_t + v_t \tag{2}$$

The system order is $p$. $\mathbf{x}_t \in \mathbb{R}^{p \times 1}$ is the state vector. $\mathbf{A} \in \mathbb{R}^{p \times p}$ and $\mathbf{c} \in \mathbb{R}^{1 \times p}$ are system parameters. The output $y_t \in \mathbb{R}$ is the speech signal at the microphone. Process and observation noises are modelled as white zero-mean Gaussian stationary noises $\mathbf{w}_t \in \mathbb{R}^{p \times 1} \sim N(\mathbf{0}, \mathbf{Q})$ and $v_t \in \mathbb{R} \sim N(0, R)$ respectively. The problem definition is to estimate parameters $\Theta = (\mathbf{A}, \mathbf{c}, \mathbf{Q}, R)$ from speech $y_t$ only.

The structure of the paper is as follows. The theory section describes EM and 4SID applied to the parameter estimation of the above SS model. The similarity of 4SID and EM motivates the use of 4SID to initialise EM. Experiments on real speech then compare 4SID with more conventional initialisation methods. The discussion then compares 4SID with EM.

# 2 Theory

## 2.1 The Expectation-Maximisation (EM) Technique

Given a sequence of $N$ observations $\mathbf{y}_{1:N}$ of a signal such as speech, the maximum likelihood estimate for the parameters is $\hat{\Theta}_{ML} = \arg \max_{\Theta} p(\mathbf{y}_{1:N}|\Theta)$. EM breaks the maximisation of this potentially difficult likelihood function down into an iterative maximisation of a simpler likelihood function, generating a new estimate $\Theta_k$ each iteration. Rewriting $p(\mathbf{y}_{1:N}|\Theta)$ in terms of a hidden state sequence $\mathbf{x}_{1:N}$, and taking expectations over $p(\mathbf{x}_{1:N}|\mathbf{y}_{1:N}, \Theta_k)$

$$\log p(\mathbf{y}_{1:N}|\Theta) = \log p(\mathbf{x}_{1:N}, \mathbf{y}_{1:N}|\Theta) - \log p(\mathbf{x}_{1:N}|\mathbf{y}_{1:N}, \Theta) \tag{3}$$
$$\log p(\mathbf{y}_{1:N}|\Theta) = E_k[\log p(\mathbf{x}_{1:N}, \mathbf{y}_{1:N}|\Theta)] - E_k[\log p(\mathbf{x}_{1:N}|\mathbf{y}_{1:N}, \Theta)] \tag{4}$$

Iterative maximisation of the first expectation in equation 4 guarantees an increase in $\log p(\mathbf{y}_{1:N}|\Theta)$.

$$\Theta_{k+1} = \arg \max_{\Theta} E_k[\log p(\mathbf{x}_{1:N}, \mathbf{y}_{1:N}|\Theta_k)] \tag{5}$$

This converges to a local or global maximum depending on the initial parameter estimate $\Theta_0$. Refer to [8] for more details. EM can thus be applied to the stochastic state space

model of equations 1 and 2 to determine optimal parameters $\Theta$. An explanation is given in [3]. The EM algorithm applied to the SS system consists of two stages per iteration. Firstly, given current parameter estimates, states are estimated using a Kalman smoother. Secondly, given these states, new parameters are estimated by maximising the expected log likelihood function. We employ the Rauch-Tung-Striebel formulation of the Kalman smoother [2].

## 2.2  The State-Space Model

Equations 1 and 2 can be cast in block matrix form and are termed the state sequence and block output equations respectively [10]. Note that the use of blocking and fixed-length signals applies restrictions to the general model in section 1. $i > p$ is the block size.

$$\mathbf{X}_{i+1,i+j} = \mathbf{A}^i\mathbf{X}_{1,j} + \mathbf{\Delta}_i^w\mathbf{W}_{1|i} \tag{6}$$

$$\mathbf{Y}_{1|i} = \mathbf{\Gamma}_i\mathbf{X}_{1,j} + \mathbf{H}_i^w\mathbf{W}_{1|i} + \mathbf{V}_{1|i} \tag{7}$$

$\mathbf{X}_{i+1,i+j}$ is a state sequence matrix; its columns are the state vectors from time $(i+1)$ to $(i+j)$. $\mathbf{X}_{1,j}$ is similarly defined. $\mathbf{Y}_{1|i}$ is a Hankel matrix of outputs from time 1 to $(i+j-1)$. $\mathbf{W}$ and $\mathbf{V}$ are similarly defined. $\mathbf{\Delta}_i^w$ is a reversed extended controllability-type matrix, $\mathbf{\Gamma}_i$ is the extended observability matrix and $\mathbf{H}_i^w$ is a Toeplitz matrix. These are all defined below where $\mathbf{I}^{p \times p}$ is an identity matrix.

$$\mathbf{X}_{1,j} \overset{\text{def}}{=} [\mathbf{x}_1 \ \mathbf{x}_2 \ \mathbf{x}_3 \ \dots \ \mathbf{x}_j] \qquad \mathbf{\Gamma}_i \overset{\text{def}}{=} \begin{bmatrix} \mathbf{c} \\ \mathbf{cA} \\ \vdots \\ \mathbf{cA}^{i-1} \end{bmatrix}$$

$$\mathbf{\Delta}_i^w \overset{\text{def}}{=} [\mathbf{A}^{i-1} \ \mathbf{A}^{i-2} \ \dots \ \mathbf{I}]$$

$$\mathbf{Y}_{1|i} \overset{\text{def}}{=} \begin{bmatrix} y_1 & y_2 & \cdots & y_j \\ y_2 & y_3 & \cdots & y_{j+1} \\ \vdots & \vdots & & \vdots \\ y_i & y_{i+1} & \cdots & y_{i+j-1} \end{bmatrix} \qquad \mathbf{H}_i^w \overset{\text{def}}{=} \begin{bmatrix} 0 & & & 0 \\ \mathbf{c} & 0 & & \\ \vdots & \ddots & \ddots & \\ \mathbf{cA}^{i-2} & \cdots & \mathbf{c} & 0 \end{bmatrix}$$

A sequence of outputs can be separated into two block output equations containing *past* and *future* outputs denoted with subscripts $p$ and $f$ respectively. With $\mathbf{Y}_p \overset{def}{=} \mathbf{Y}_{1|i}$, $\mathbf{Y}_f \overset{def}{=} \mathbf{Y}_{i+1|2i}$ and similarly for $\mathbf{W}$ and $\mathbf{V}$, and $\mathbf{X}_p \overset{def}{=} \mathbf{X}_{1,j}$ and $\mathbf{X}_f \overset{def}{=} \mathbf{X}_{i+1,i+j}$, past and future are related by the equations

$$\mathbf{X}_f = \mathbf{A}^i\mathbf{X}_p + \mathbf{\Delta}_i^w\mathbf{W}_p \tag{8}$$

$$\mathbf{Y}_p = \mathbf{\Gamma}_i\mathbf{X}_p + \mathbf{H}_i^w\mathbf{W}_p + \mathbf{V}_p \tag{9}$$

$$\mathbf{Y}_f = \mathbf{\Gamma}_i\mathbf{X}_f + \mathbf{H}_i^w\mathbf{W}_f + \mathbf{V}_f \tag{10}$$

## 2.3  Subspace State Space System Identification (4SID) Techniques

Comments throughout this section on 4SID are largely taken from the work of Van Overschee and De Moor [10]. 4SID methods are related to instrumental variable (IV) methods [11]. 4SID algorithms are composed of two stages. Stage one involves the low-rank approximation and estimation of the extended observability matrix directly from the output

data. For example, consider the future output block equation 10. $\mathbf{Y}_f$ undergoes an orthogonal projection onto the row space of $\mathbf{Y}_p$. This is denoted by $\mathbf{Y}_f/\mathcal{Y}_p = \mathbf{Y}_f \mathbf{Y}_p^T (\mathbf{Y}_p \mathbf{Y}_p^T)^\dagger \mathbf{Y}_p$, where $^\dagger$ is the Moore-Penrose inverse.

$$
\begin{aligned}
\mathbf{Y}_f/\mathcal{Y}_p &= \mathbf{\Gamma}_i \mathbf{X}_f/\mathcal{Y}_p + \mathbf{H}_i^w \mathbf{W}_f/\mathcal{Y}_p + \mathbf{V}_f/\mathcal{Y}_p \\
\mathbf{Y}_f/\mathcal{Y}_p &= \mathbf{\Gamma}_i \mathbf{X}_f/\mathcal{Y}_p
\end{aligned}
\tag{11}
$$

Stage two involves estimation of system parameters. The singular value decomposition of $\mathbf{Y}_f/\mathcal{Y}_p$ allows the observability and state sequence matrices to be estimated to within a similarity transform from the column and row spaces respectively. From these two matrices, system parameters $(\mathbf{A}, \mathbf{c}, \mathbf{Q}, R)$ can be determined by least-squares.

There are two interesting comments. Firstly, the orthogonal projection from stage one coincides with a minimum error between true data $\mathbf{Y}_f$ and its linear prediction from $\mathbf{Y}_p$ in the Frobenius norm. Greater flexibility is obtained by weighting the projection with matrices $\mathbf{W}_1$ and $\mathbf{W}_2$ and analysing this: $\mathbf{W}_1(\mathbf{Y}_f/\mathcal{Y}_p)\mathbf{W}_2$. 4SID and IV methods differ with respect to these weighting matrices. Weighting is similar to prefiltering the observations prior to analysis to preferentially weight some frequency domain, as is common in identification theory [6]. Secondly, the state estimates from stage two can be considered as outputs from a parallel bank of Kalman filters, each one estimating a state from the previous $i$ observations, and initialised using zero conditions.

The particular subspace algorithm and software used in this paper is the *sto_pos* algorithm as detailed in [10]. Although this algorithm introduces a small bias into some of the parameter estimates, it guarantees positive realness of the covariance sequence, which in turn guarantees the definition of a forward innovations model.

## 3 Experiments

Experiments are conducted on the phrase *"in arithmetic"*, spoken by an adult male. The speech waveform is obtained from the Eurom 0 database [4] and sampled at 16 kHz. The speech waveform is divided into fixed-length, overlapping frames, the mean is subtracted and then a hamming window is applied. Frames are 15 ms in duration, shifted 7.5 ms each frame. Speech is modelled as detailed in section 1. All models are order 8. A frame is assumed silent and no analysis done when the mean energy per sample is less than an empirically defined threshold.

For the EM algorithm, a modified version of the software in [3] is used. The initial state vector and covariance matrix are set to zero and identity respectively, and 50 iterations are applied. $\mathbf{Q}$ is updated by taking its diagonal only in the M-step for numerical stability (see [3]).

In these experiments, three schemes are compared at initialising parameters for the EM algorithm, that is the estimation of $\Theta_0$. These schemes are compared in terms of their formant trajectories relative to the spectrogram and their likelihoods. The three schemes are

- **4SID**. This is the subspace method in section 2.3 with block size 16.
- **ARMA**. This estimates $\Theta_0$ using the customised Matlab *armax* function[1], which models the speech waveform as an autoregressive moving average (ARMA) process, with order 8 polynomials.

- **AR(1)**. This uses a simplistic method, and models the speech waveform as a first order autoregressive (AR) process with some randomness introduced into the estimation. It still initialises all parameters fully[2].

Results are shown in Figures 1 and 2. Figure 1 shows the speech waveform, spectrogram and formant trajectories for EM with all three initialisation schemes. Here formant frequencies are derived from the phase of the positive phase eigenvalues of **A** after 50 iterations of EM. Comparison with the spectrogram shows that for this order 8 model, 4SID-EM produces best formant trajectories. Figure 2 shows mean average plots of likelihood against EM iteration number for each initialisation scheme. 4SID-EM gives greater likelihoods than ARMA-EM and AR(1)-EM. The difference in formant trajectories between subspace-EM and ARMA-EM despite the high likelihoods, demonstrates the multi-modality of the likelihood function. For AR(1)-EM, a few frames were not estimated due to numerical instability.

## 4   Discussion

Both the 4SID and EM algorithms employ similar methodologies: states are first estimated using a Kalman device, and then these states are used to estimate system parameters according to similar criteria. However in EM, states are estimated using past, present and future observations with a Kalman smoother; system parameters are then estimated using maximum likelihood (ML). Whereas in 4SID, states are estimated using the previous $i$ observations only with non-steady state Kalman filters. System parameters are then estimated using least-squares (LS) subject to a positive realness constraint for the covariance sequence. Refer also to [5] for a similar comparison.

4SID algorithms are sub-optimal for three reasons. Firstly, states are estimated using only partial observations sequences. Secondly, the LS criterion is only an approximation to the ML criterion. Thirdly, the positive realness constraint introduces bias. A positive realness constraint is necessary due to a finite amount of data and any lacking in the SS model. For this reason, 4SID methods are used to initialise rather than replace EM in these experiments.

4SID methods also have some advantages. Firstly, they are linear and non-iterative, and do not suffer from the disadvantages typical of iterative algorithms (including EM) such as sensitivity to initial conditions, convergence to local minima, and the definition of convergence criteria. Secondly, they require little prior parameterisation except the definition of the system order, which can be determined *in situ* from observation of the singular values of the orthogonal projection. Thirdly, the use of the SVD gives numerical robustness to the algorithms. Fourthly, they have higher frequency resolution than prediction error minimisation methods such as ARMA and AR [1].

## 5   Conclusions

4SID methods can be used to initialise EM giving better formant tracks, higher likelihoods and better frequency resolution than more conventional initialisation methods. In the future we hope to compare 4SID methods with EM in a principled probabilistic manner, investigate weighting matrices further, and apply these methods to speech enhancement. Further work is done by Smith et al. in [9], and similar work done by Grivel et al. in [5].

*Acknowledgements*
We are grateful for the use of 4SID software supplied with [10] and the EM software of

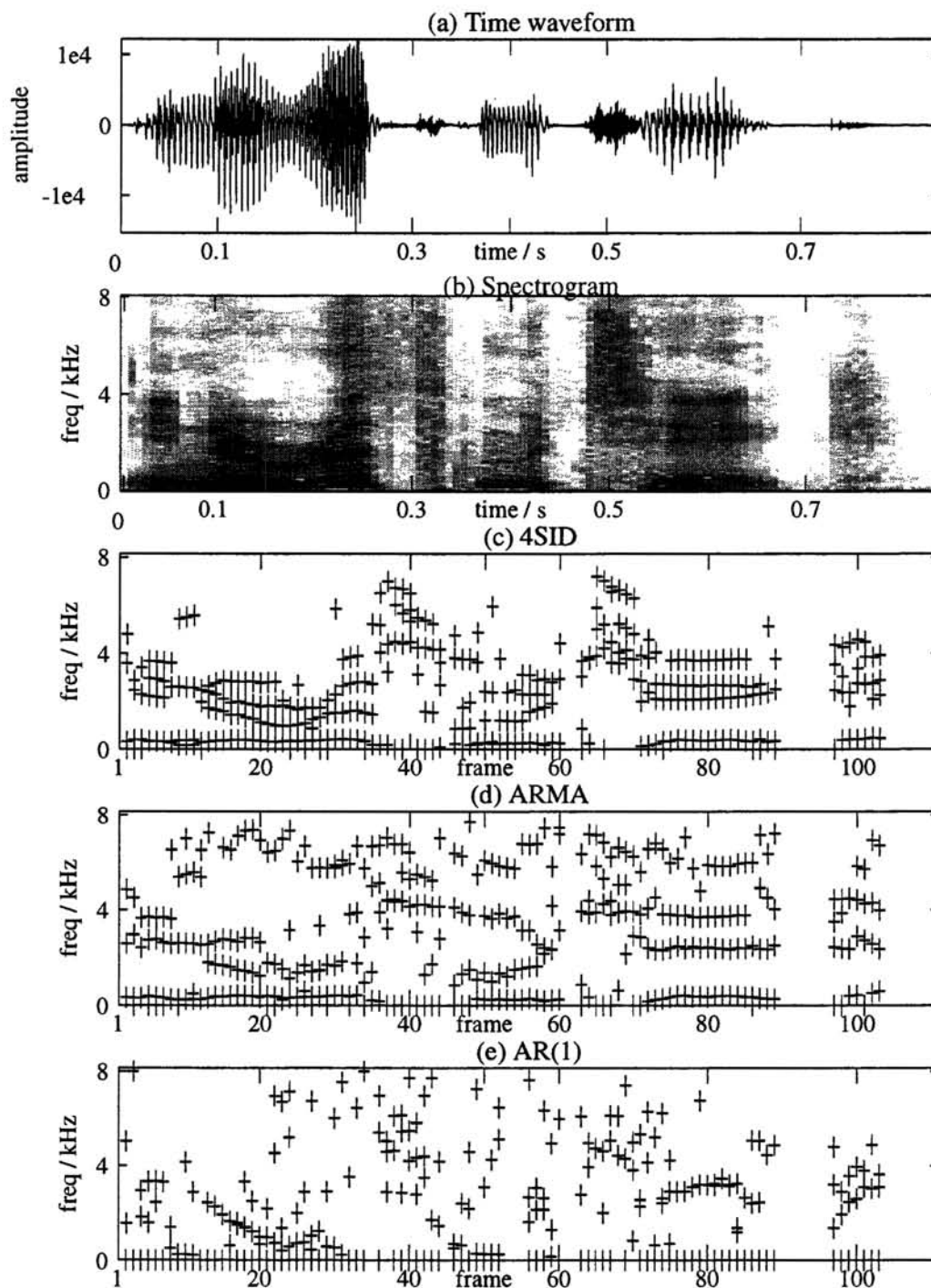

Figure 1: (a) Time waveform and (b) spectrogram for "*in arithmetic*". Formant trajectories are estimated using EM and a SS model initialised with three different schemes: (d) 4SID, (e) ARMA and (f) AR(1).

Zoubin Ghahramani [3]. Gavin Smith is supported by the Schiff Foundation, Cambridge University. At the time of writing, Nando de Freitas was supported by two University of the Witwatersrand Merit Scholarships, a Foundation for Research Development Scholarship (South Africa), an ORS award and a Trinity College External Research Studentship (Cambridge).

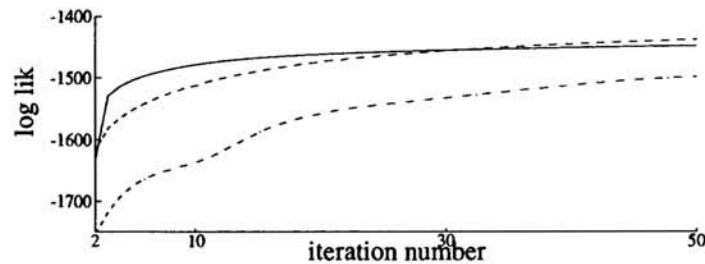

Figure 2: Likelihood convergence plots for EM and the SS model initialised with 4SID [- -], ARMA [–] and AR(1) [-.] for the experiments in Figure 1. Plots are the mean average over all frames where analysed.

# 6 References

[1] Arun, K.S. & Kung, S.Y. (1990) Balanced Approximation of Stochastic Systems. SIAM Journal on Matrix Analysis and Applications, vol. 11, no. 1, pp. 42–68.

[2] Gelb, A. ed., (1974) *Applied Optimal Estimation.* Cambridge, MA: MIT Press.

[3] Ghahramani, Z. & Hinton, G. (1996) Parameter Estimation for Linear Dynamical Systems, Tech. rep. CRG-TR-96-2, Dept. of Computer Science, Univ. of Toronto. Software at *www.gatsby.ucl.ac.uk/˜zoubin/software.html.*

[4] Grice, M. & Barry, W. (1989) Multi-lingual Speech Input/Output: Assessment, Methodology and Standardization, Tech. rep., University College, London, ESPRIT Project 1541 (SAM), extension phase final report.

[5] Grivel, E., Gabrea, M. & Najim, M. (1999) Subspace State Space Model Identification For Speech Enhancement, Paper 1622, ICASSP'99.

[6] Ljung, L. (1987) *System Identification: Theory for the User.* Englewood Cliffs, NJ: Prentice-Hall, Inc.

[7] Ljung, L. (1991) *System Identification Toolbox For Use With MatLab.* 24 Prime Park Way, Natrick, MA, USA: The MathWorks, Inc.

[8] McLachlan, G.J. & Krishnan, T. (1997) *The EM Algorithm and Extensions.* John Wiley and Sons Inc.

[9] Smith, G.A. & Robinson, A.J. & Niranjan, M. (2000) A Comparison Between the EM and Subspace Algorithms for the Time-Invariant Linear Dynamical System. Tech. rep. CUED/F-INFENG/TR.366, Engineering Dept., Cambridge Univ., UK.

[10] Van Overschee, P. & De Moor, B. (1996) *Subspace Identification for Linear Systems: Theory, Implementation, Applications.* Dordrecht, Netherlands: Kluwer Academic Publishers.

[11] Viberg, M. & Wahlberg, B. & Ottersten, B. (1997) Analysis of State Space System Identification Methods Based on Instrumental Variables and Subspace Fitting. Automatica, vol. 33, no. 9, pp. 1603–1616.

## Footnotes

[1] Work done while in Cambridge Engineering Dept., UK.

[1]*armax* minimises a robustified quadratic prediction error criterion using an iterative Gauss-Newton algorithm, initialised using a four-stage least-squares instrumental variables algorithm [7].

[2]Presented in the software in [3], this method is best used when the dimensions of the state space and observations are the same.
